# Kernel Maximum Entropy Data Transformation and an Enhanced Spectral Clustering Algorithm

**Robert Jenssen**[1,*] **Torbjørn Eltoft**[1], **Mark Girolami**[2] **and Deniz Erdogmus**[3]

[1] Department of Physics and Technology, University of Tromsø, Norway
[2] Department of Computing Science, University of Glasgow, Scotland
[3]Department of Computer Science and Engineering, Oregon Health and Science University, USA

### Abstract

We propose a new kernel-based data transformation technique. It is founded on the principle of maximum entropy (MaxEnt) preservation, hence named kernel MaxEnt. The key measure is Renyi's entropy estimated via Parzen windowing. We show that kernel MaxEnt is based on eigenvectors, and is in that sense similar to kernel PCA, but may produce strikingly different transformed data sets. An enhanced spectral clustering algorithm is proposed, by replacing kernel PCA by kernel MaxEnt as an intermediate step. This has a major impact on performance.

## 1 Introduction

Data transformation is of fundamental importance in machine learning, and may greatly improve and simplify tasks such as clustering. Some of the most well-known approaches to data transformation are based on eigenvectors of certain matrices. Traditional techniques include principal component analysis (PCA) and classical multidimensional scaling. These are linear methods. Recent advanced non-linear techniques include locally-linear embedding [1] and isometric mapping [2]. Of special interest to this paper is kernel PCA [3], a member of the kernel-based methods [4].

Recently, it has been shown that there is a close connection between the kernel methods and information theoretic learning [5, 6, 7, 8]. We propose a new kernel-based data transformation technique based on the idea of maximum entropy preservation. The new method, named kernel MaxEnt, is based on Renyi's quadratic entropy estimated via Parzen windowing. The data transformation is obtained using eigenvectors of the data affinity matrix. These eigenvectors are in general not the same as those used in kernel PCA. We show that kernel MaxEnt may produce strikingly different transformed data sets than kernel PCA. We propose an enhanced spectral clustering algorithm, by replacing kernel PCA by kernel MaxEnt as an intermediate step. This seemingly minor adjustment has a huge impact on the performance of the algorithm.

This paper is organized as follows. In section 2, we briefly review kernel PCA. Section 3 is devoted to the kernel MaxEnt method. Some illustrations are given in section 4. The enhanced spectral clustering is discussed in section 5. Finally, we conclude the paper in section 6.

## 2 Kernel PCA

PCA is a linear data transformation technique based on the eigenvalues and eigenvectors of the $(d \times d)$ data correlation matrix, where $d$ is the data dimensionality. A dimensionality reduction from $d$ to $l < d$ is obtained by projecting a data point onto a subspace spanned by the eigenvectors (principal axes) corresponding to the $l$ largest eigenvalues. It is well-known that this data

transformation preserves the maximum amount of *variance* in the $l$-dimensional data compared to the original $d$-dimensional data. Schölkopf et al. [3] proposed a non-linear extension, by performing PCA implicitly in a kernel feature space which is non-linearly related to the input space via the mapping $\mathbf{x}_i \rightarrow \mathbf{\Phi}(\mathbf{x}_i)$, $i = 1, \ldots, N$. Using the kernel-trick to compute inner-products, $k(\mathbf{x}_i, \mathbf{x}_j) = \langle \mathbf{\Phi}(\mathbf{x}_i), \mathbf{\Phi}(\mathbf{x}_j) \rangle$, it was shown that the eigenvalue problem in terms of the feature space correlation matrix is reduced to an eigenvalue problem in terms of the kernel matrix $\mathbf{K_x}$, where element $(i, j)$ of $\mathbf{K_x}$ equals $k(\mathbf{x}_i, \mathbf{x}_j)$, $i, j = 1, \ldots, N$. This matrix can be eigendecomposed as $\mathbf{K_x} = \mathbf{E} \mathbf{D} \mathbf{E}^T$, where $\mathbf{D}$ is a diagonal matrix storing the eigenvalues in descending order, and $\mathbf{E}$ is a matrix with the eigenvectors as columns. Let $\mathbf{\Phi}_{pca}$ be a matrix where each column corresponds to the PCA projection of the data points $\mathbf{\Phi}(\mathbf{x}_i)$, $i = 1, \ldots, N$, onto the subspace spanned by the $l$ largest kernel space principal axes. Then, $\mathbf{\Phi}_{pca} = \mathbf{D}_l^{\frac{1}{2}} \mathbf{E}_l^T$, where the $(l \times l)$ matrix $\mathbf{D}_l$ stores the $l$ largest eigenvalues, and the $(N \times l)$ matrix $\mathbf{E}_l$ stores the corresponding eigenvectors. This is the kernel PCA transformed data set [1]. Kernel PCA thus preserves *variance* in terms of the *kernel induced feature space*.

However, kernel PCA is not easily interpreted in terms of the *input space* data set. How does variance preservation in the kernel feature space correspond to an operation on the input space data set? To the best of our knowledge, there are no such intuitive interpretations of kernel PCA. In the next section, we introduce kernel MaxEnt, which we show is related to kernel PCA. However, kernel MaxEnt may be *interpreted* in terms of the input space, and will in general perform a different projection in the kernel space.

## 3  Kernel MaxEnt

The Renyi quadratic entropy is given by $H_2(\mathbf{x}) = -\log \int f^2(\mathbf{x}) d\mathbf{x}$ [9], where $f(\mathbf{x})$ is the density associated with the random variable $\mathbf{X}$. A $d$-dimensional data set $\mathbf{x}_i, i = 1, \ldots, N$, generated from $f(\mathbf{x})$, is assumed available. A non-parametric estimator for $H_2(\mathbf{x})$ is obtained by replacing the actual pdf by its Parzen window estimator, given by [10]

$$\hat{f}(\mathbf{x}) = \frac{1}{N} \sum_{i=1}^{N} W_\sigma(\mathbf{x}, \mathbf{x}_i), \quad W_\sigma(\mathbf{x}, \mathbf{x}_i) = \frac{1}{(2\pi\sigma^2)^{\frac{d}{2}}} \exp \left\{ -\frac{||\mathbf{x} - \mathbf{x}_i||^2}{2\sigma^2} \right\}. \quad (1)$$

The Parzen window need not be Gaussian, but it must be a density itself. The following derivation assumes a Gaussian window. Non-Gaussian windows are easily incorporated. Hence, we obtain

$$\hat{H}_2(\mathbf{x}) = -\log \int \hat{f}^2(\mathbf{x}) d\mathbf{x} = -\log \frac{1}{N^2} \sum_{i=1}^{N} \sum_{j=1}^{N} \int W_\sigma(\mathbf{x}, \mathbf{x}_i) W_\sigma(\mathbf{x}, \mathbf{x}_j) d\mathbf{x}$$

$$= -\log \frac{1}{N^2} \sum_{i=1}^{N} \sum_{j=1}^{N} W_{\sqrt{2}\sigma}(\mathbf{x}_i, \mathbf{x}_j), \quad (2)$$

where in the last step the convolution theorem for Gaussian functions has been employed.

For notational simplicity, we denote $W_{\sqrt{2}\sigma}(\mathbf{x}_i, \mathbf{x}_j)$ as $k(\mathbf{x}_i, \mathbf{x}_j)$. Note that since $W_{\sqrt{2}\sigma}(\cdot, \cdot)$ is a Gaussian function, it is also a Mercer kernel, and so is $k(\cdot, \cdot)$. In the following, we construct the kernel matrix $\mathbf{K_x}$, such that element $(i, j)$ of $\mathbf{K_x}$ equals $k(\mathbf{x}_i, \mathbf{x}_j)$, $i, j = 1, \ldots, N$.

It is easily shown that the Renyi quadratic entropy may be expressed compactly in terms of the kernel matrix as $\hat{H}_2(\mathbf{x}) = -\log \frac{1}{N^2} \mathbf{1}^T \mathbf{K_x} \mathbf{1}$, where $\mathbf{1}$ is a $(N \times 1)$ ones-vector. Since the logarithm is a monotonic function, we will in the remainder of this paper focus on the quantity $V(\mathbf{x}) = \frac{1}{N^2} \mathbf{1}^T \mathbf{K_x} \mathbf{1}$. It is thus clear that all the information regarding the Renyi entropy resides in the kernel matrix $\mathbf{K_x}$. Hence, the kernel matrix is the input space quantity of interest in this paper.

A well-known *input space* data transformation principle is founded on the idea of maximum entropy (MaxEnt), usually defined in terms of minimum model complexity. In this paper, we define MaxEnt differently, as a mapping $\mathbf{X} \rightarrow \mathbf{Y}$, such that the entropy associated with $\mathbf{Y}$ is *maximally similar*

to the entropy of $\mathbf{X}$. Since we are concerned with Renyi's entropy, it is therefore clear that such a data mapping results in a $V(\mathbf{y}) = \frac{1}{N^2}\mathbf{1}^T\mathbf{K_y}\mathbf{1}$, in terms of the $\mathbf{Y}$ data set, which should be as close as possible to $V(\mathbf{x}) = \frac{1}{N^2}\mathbf{1}^T\mathbf{K_x}\mathbf{1}$. This means that the kernel matrix $\mathbf{K_y}$ must be maximally similar to $\mathbf{K_x}$ in some sense. Since our input space quantity of concern is the kernel matrix, we are only *implicitly* concerned with the $\mathbf{Y}$ data set (we do not actually want to obtain $\mathbf{Y}$, nor is the dimensionality of interest).

The kernel matrix can be decomposed as $\mathbf{K_x} = \mathbf{E}\mathbf{D}\mathbf{E}^T$. The kernel matrix is at the same time an inner-product matrix in the Mercer kernel induced feature space. Let $\mathbf{\Phi_x}$ be a matrix such that each column represents an *approximation* to the corresponding kernel feature space data point in the set $\mathbf{\Phi}(\mathbf{x}_1), \dots, \mathbf{\Phi}(\mathbf{x}_N)$. An approximation which preserves inner-products is given by $\mathbf{\Phi_x} = \mathbf{D}^{\frac{1}{2}}\mathbf{E}^T$, since then $\mathbf{K_x} = \mathbf{\Phi_x}^T\mathbf{\Phi_x} = \mathbf{E}\mathbf{D}\mathbf{E}^T$. Note that $\mathbf{\Phi_x} = \mathbf{D}^{\frac{1}{2}}\mathbf{E}^T$ is the projection onto *all* the principal axes in the Mercer kernel feature space, hence defining a $N$-dimensional data set.

We now describe a dimensionality reduction in the Mercer kernel space, obtaining the $k$-dimensional $\mathbf{\Phi_y}$ from $\mathbf{\Phi_x}$, yielding $\mathbf{K_y} = \mathbf{\Phi_y}^T\mathbf{\Phi_y}$ such that $V(\mathbf{y}) \approx V(\mathbf{x})$. Notice that we may rewrite $V(\mathbf{x})$ as follows [8]

$$V(\mathbf{x}) = \frac{1}{N^2}\sum_{i=1}^{N}\lambda_i(\mathbf{1}^T\mathbf{e}_i)^2 = \frac{1}{N^2}\sum_{i=1}^{N}\lambda_i\gamma_i^2, \qquad (3)$$

where $\mathbf{e}_i$ is the eigenvector corresponding to the $i$'th column of $\mathbf{K_x}$, and $\mathbf{1}^T\mathbf{e}_i = \gamma_i$. We also assume that the products $\lambda_i\gamma_i^2$ have been sorted in decreasing order, such that $\lambda_1\gamma_1^2 \geq \dots \geq \lambda_N\gamma_N^2$.

If we are to approximate $V(\mathbf{x})$ using only $k$ terms (eigenvalues/eigenvectors) of the sum Eq. (3), we must use the $k$ first terms in order to achieve minimum approximation error. This corresponds to using the $k$ largest $\lambda_i\gamma_i^2$. Let us define the data set $\mathbf{\Phi_y} = \mathbf{D}_k^{\frac{1}{2}}\mathbf{E}_k^T$, using the $k$ eigenvalues and eigenvectors of $\mathbf{K_x}$ corresponding to the $k$ largest products $\lambda_i\gamma_i^2$. Hence, $\mathbf{K_y} = \mathbf{\Phi_y}^T\mathbf{\Phi_y} = \mathbf{E}_k\mathbf{D}_k^{\frac{1}{2}}\mathbf{D}_k^{\frac{1}{2}}\mathbf{E}_k^T = \mathbf{E}_k\mathbf{D}_k\mathbf{E}_k^T$, and

$$V(\mathbf{y}) = \frac{1}{N^2}\sum_{i=1}^{k}\lambda_i\gamma_i^2 = \frac{1}{N^2}\mathbf{1}^T\mathbf{K_y}\mathbf{1}, \qquad (4)$$

the *best* approximation to the entropy estimate $V(\mathbf{x})$ using $k$ eigenvalues and eigenvectors. We thus refer to the mapping $\mathbf{\Phi_y} = \mathbf{D}_k^{\frac{1}{2}}\mathbf{E}_k^T$ as a *maximum entropy* data transformation in a Mercer kernel feature space. Note that this is *not* the same as the PCA dimensionality reduction in Mercer kernel feature space, which is defined as $\mathbf{\Phi}_{pca} = \mathbf{D}_l^{\frac{1}{2}}\mathbf{E}_l^T$, using the eigenvalues and eigenvectors corresponding to the $l$ largest eigenvalues of $\mathbf{K_x}$. In terms of the eigenvectors of the kernel feature space correlation matrix, we project $\mathbf{\Phi}(\mathbf{x}_i)$ onto a subspace spanned by different eigenvectors, which is possibly *not* the most variance preserving (remember that variance in the kernel feature space data set is given by the sum of the largest eigenvalues).

The kernel MaxEnt procedure, as described above, is summarized in Table 1. It is important to realize that kernel MaxEnt outputs *two* quantities, which may be used for further data analysis. The *kernel space* output quantity is the transformed data set $\mathbf{\Phi_y} = \mathbf{D}_k^{\frac{1}{2}}\mathbf{E}_k^T$. The *input space* output quantity is the kernel matrix $\mathbf{K_y} = \mathbf{E}_k\mathbf{D}_k\mathbf{E}_k^T$, which is an approximation to the original kernel matrix $\mathbf{K_x}$.

| *Input Space* | | *Kernel Space* |
|---|---|---|
| $\mathbf{K_x} = \mathbf{E}\mathbf{D}\mathbf{E}^T$ | $\rightarrow$ | $\mathbf{\Phi_x} = \mathbf{D}^{\frac{1}{2}}\mathbf{E}^T$ |
| $\wr$ | | $\downarrow$ |
| $\mathbf{K_y} = \mathbf{E}_k\mathbf{D}_k\mathbf{E}_k^T$ | $\leftarrow$ | $\mathbf{\Phi_y} = \mathbf{D}_k^{\frac{1}{2}}\mathbf{E}_k^T$ |

Table 1. Flow of the kernel MaxEnt procedure. There are two possible outputs; the input space kernel matrix $\mathbf{K_y}$, and the kernel space data set $\mathbf{\Phi_y}$.

### 3.1 Interpretation in Terms of Cost Function Minimization

Kernel MaxEnt produces a new kernel matrix $\mathbf{K_y} = \mathbf{E}_k \mathbf{D}_k \mathbf{E}_k^T$, such that the sum of the elements of $\mathbf{K_y}$ is maximally equal to the sum of the elements of $\mathbf{K_x}$. Hence, kernel MaxEnt picks eigenvectors and eigenvalues in order to minimize the cost function $\mathbf{1}^T(\mathbf{K_x} - \mathbf{K_y})\mathbf{1}$. On the other hand, it is well known that the kernel PCA matrix $\mathbf{K}_{pca} = \mathbf{E}_l \mathbf{D}_l \mathbf{E}_l^T$, based on the $l$ largest eigenvalues, minimizes the Frobenius norm of $(\mathbf{K_x} - \mathbf{K}_{pca})$, that is $\mathbf{1}^T(\mathbf{K_x} - \mathbf{K}_{pca}).^2\mathbf{1}$ (where $\mathbf{A}.^2$ denotes elementwise squaring of matrix $\mathbf{A}$.)

### 3.2 Kernel MaxEnt Eigenvectors Reveal Cluster Structure

Under "ideal" circumstances, kernel MaxEnt and kernel PCA yield the same result, as shown in the following. Assume that the data consists of $C = 2$ different maximally compact subsets, such that $k(\mathbf{x}_i, \mathbf{x}_j) = 1$ for $\mathbf{x}_i$ and $\mathbf{x}_j$ in the same subset, and $k(\mathbf{x}_i, \mathbf{x}_j) = 0$ for $\mathbf{x}_i$ and $\mathbf{x}_j$ in different subsets (point clusters). Assume that subset one consists of $N_1$ data points, and subset two consists of $N_2$ data points. Hence, $N = N_1 + N_2$ and we assume $N_1 \geq N_2$. Then

$$\mathbf{K} = \left[ \begin{array}{cc} \underline{\mathbf{1}}^{N_1 \times N_1} & \underline{\mathbf{0}}^{N_1 \times N_2} \\ \underline{\mathbf{0}}^{N_2 \times N_1} & \underline{\mathbf{1}}^{N_2 \times N_2} \end{array} \right], \ \mathbf{E} = \left[ \begin{array}{cc} \frac{1}{\sqrt{N_1}} \mathbf{1}_{N_1} & \mathbf{0}_{N_1} \\ \mathbf{0}_{N_2} & \frac{1}{\sqrt{N_2}} \mathbf{1}_{N_2} \end{array} \right], \tag{5}$$

where $\underline{\mathbf{1}}^{M \times M}$ $(\underline{\mathbf{0}}^{M \times M})$ is the $(M \times M)$ all ones (zero) matrix and $\mathbf{D} = diag(N_1, N_2)$. Hence, a data point $\mathbf{x}_i$ in subgroup one will be represented by $\mathbf{x}_i \rightarrow [1 \ 0]^T$ and a data point $\mathbf{x}_j$ in subgroup two will be represented by $\mathbf{x}_j \rightarrow [0 \ 1]^T$ both using $\mathbf{\Phi_y}$ and $\mathbf{\Phi}_{pca}$. (see also [11] for a related analysis). Thus, kernel MaxEnt and kernel PCA yield the same data mapping, where each subgroup is mapped into mutually orthogonal points in the kernel space (the clusters were points also in the input space, but not necessarily orthogonal). Hence, in the "ideal" case, the clusters in the transformed data set is spread by 90 degrees *angles*. Also, the eigenvectors carry all necessary information about the cluster structure (cluster memberships can be assigned by a proper thresholding). This kind of "ideal" analysis has been used as a justification for the kernel PCA mapping, where the mapping is based on the $C$ largest eigenvalues/eigenvectors. Such a situation will correspond to maximally concentrated eigenvalues of the kernel matrix.

In practice, however, there will be more than $C$ non-zero eigenvalues, not necessarily concentrated, and corresponding eigenvectors, because there will be no such "ideal" situation. Shawe-Taylor and Cristianini [4] note that kernel PCA can only detect stable patterns if the eigenvalues are concentrated. In practice, the first $C$ eigenvectors may not necessarily be those which carry most information about the clustering structure of the data set. However, kernel MaxEnt will seek to pick those eigenvectors with the blockwise structure corresponding to cluster groupings, because this will make the sum of the elements in $\mathbf{K_y}$ as close as possible to the sum of the elements of $\mathbf{K_x}$. Some illustrations of this property follow in the next section.

### 3.3 Parzen Window Size Selection

The Renyi entropy estimate is directly connected to Parzen windowing. In theory, therefore, an appropriate window, or kernel, size, corresponds to an appropriate density estimate. Parzen window size selection has been thoroughly studied in statistics [12]. Many reliable data-driven methods exist, especially for data sets of low to moderate dimensionality. Silverman's rule [12] is one of the simplest, given by $\hat{\sigma} = \sigma_X \left[ \frac{4}{(2d+1)N} \right]^{\frac{1}{d+4}}$, where $\sigma_X^2 = d^{-1} \sum_i \mathbf{\Sigma}_{X_{ii}}$, and $\mathbf{\Sigma}_{X_{ii}}$ are the diagonal elements of the sample covariance matrix. Unless otherwise stated, the window size is determined using this rule.

## 4 Illustrations

Fig. 1 (a) shows a ring-shaped data set consisting of $C = 3$ clusters (marked with different symbols for clarity). The vertical lines in (b) show the 10 largest eigenvalues (normalized). The largest eigenvalue is more than twice as large as the second largest. However, the values of the remaining eigenvalues are not significantly different. The bars in (b) shows the entropy terms $\lambda_i \gamma_i^2$ (normalized)

corresponding to these largest eigenvalues. Note that the entropy terms corresponding to the first, fourth and seventh eigenvalues are significantly larger than the rest. This means that kernel MaxEnt is based on the first, fourth and seventh eigenvalue/eigenvector pair (yielding a 3-dimensional transformed data set). In contrast, kernel PCA is based on the eigenvalue/eigenvector pair corresponding to the three largest eigenvalues. In (c) the kernel MaxEnt data transformation is shown. Note that the clusters are located along different lines radially from the origin (illustrated by the lines in the figure). These lines are almost orthogonal to each other, hence approximating what would be expected in the "ideal" case. The kernel PCA data transformation is shown in (d). This data set is significantly different. In fact, the mean vectors of the clusters in the kernel PCA representation are *not* spread angularly. In (e), the first eight eigenvectors are shown. The original data set is ordered, such that the first 63 elements correspond to the innermost ring, the next 126 elements correspond to the ring in the middle, and the final 126 elements correspond to the outermost ring. Observe how eigenvectors one, four and seven are those which carry information about the *cluster structure*, with their blockwise appearance. The kernel matrix $\mathbf{K_x}$ is shown in (f). Ideally, this should be a blockwise matrix. It is not. In (g), the kernel MaxEnt approximation $\mathbf{K_y}$ to the original kernel matrix is shown, obtained from eigenvectors one, four and seven. Note the blockwise appearance. In contrast, (g) shows the corresponding $\mathbf{K}_{pca}$. The same blockwise structure can not be observed.

Fig. 2 (a) shows a ring-shaped data set consisting of two clusters. In (b) and (c) the kernel MaxEnt (eigenvalues/eigenvectors one and five) and kernel PCA transformations are shown, respectively. Again, kernel MaxEnt produces a data set where the clusters are located along almost orthogonal lines, in contrast to kernel PCA. The same phenomenon is observed for the data set shown in (d), with the kernel MaxEnt (eigenvalues/eigenvectors one and four) and kernel PCA transformations shown in (e) and (f), respectively. In addition, (g) and (h) shows the kernel MaxEnt (eigenvalues/eigenvectors one, two and five) and kernel PCA transformations of the 16-dimensional pen-based handwritten digit recognition data set (three clusters, digits 0, 1 and 2), extracted from the UCI repository. Again, similar comments can be made. These illustrations show that kernel MaxEnt produces a different transformed data set than kernel PCA. Also, it produces a kernel matrix $\mathbf{K_y}$ having a blockwise appearance. Both the transformed data $\mathbf{\Phi_y}$ and the new kernel matrix can be utilized for further data analysis. In the following, we focus on $\mathbf{\Phi_y}$.

## 5   An Enhanced Spectral Clustering Algorithm

A recent spectral clustering algorithm [7] is based on the Cauchy-Schwarz (CS) pdf divergence measure, which is closely connected to the Renyi entropy. Let $\hat{f}_1(\mathbf{x})$ and $\hat{f}_2(\mathbf{x})$ be Parzen window estimators of the densities corresponding to two clusters. Then, an estimator for the CS measure can be expressed as [6]

$$\hat{D}(f_1, f_2) = \frac{\int \hat{f}_1(\mathbf{x})\hat{f}_2(\mathbf{x})d\mathbf{x}}{\sqrt{\int \hat{f}_1^2(\mathbf{x})d\mathbf{x} \int \hat{f}_2^2(\mathbf{x})d\mathbf{x}}} = \cos \angle(\mathbf{m}_1, \mathbf{m}_2), \tag{6}$$

where $\mathbf{m}_1$ and $\mathbf{m}_2$ are the kernel feature space mean vectors of the data points corresponding to the two clusters. Note that $\hat{D}(f_1, f_2) \in [0, 1]$, reaching its maximum value if $\mathbf{m}_1 = \mathbf{m}_2$ ($\hat{f}_1(\mathbf{x}) = \hat{f}_2(\mathbf{x})$), and its minimum value if the two vectors (densities) are orthogonal. The measure can easily be extended to more than two pdfs. The clustering is based on computing the cosine of the *angle* between a data point and the mean vector $\mathbf{m}_i$ of each cluster $\omega_i$, $i = 1, \ldots, C$, for then to assign the data point to the cluster corresponding to the maximum value. This procedure minimizes the CS measure as defined above. Kernel PCA was used for representing the data in the kernel feature space. As an illustration of the utility of kernel MaxEnt, we here replace kernel PCA by kernel MaxEnt. This adjustment has a major impact on the performance. The algorithm thus has the following steps: 1) Use some data-driven method from statistics to determine the Parzen window size. 2) Compute the kernel matrix $\mathbf{K_x}$. 3) Obtain a $C$-dimensional kernel feature space representation using kernel MaxEnt. 4) Initialize mean vectors. 5) For all data points: $\mathbf{x}_t \rightarrow \omega_i : \max_i \cos \angle(\mathbf{\Phi}(\mathbf{x}_t), \mathbf{m}_i)$. 6) Update mean vectors. 7) Repeat steps 5-7 until convergence. For further details (like mean vector initialization etc.) we refer to [7].

Fig. 3 (a) shows the clustering performance in terms of the percentage of correct labeling for the data set shown in Fig. 2 (d). There are three curves: Our spectral clustering algorithm using kernel MaxEnt (marked by the circle-symbol), and kernel PCA (star-symbol), and in addition we compare

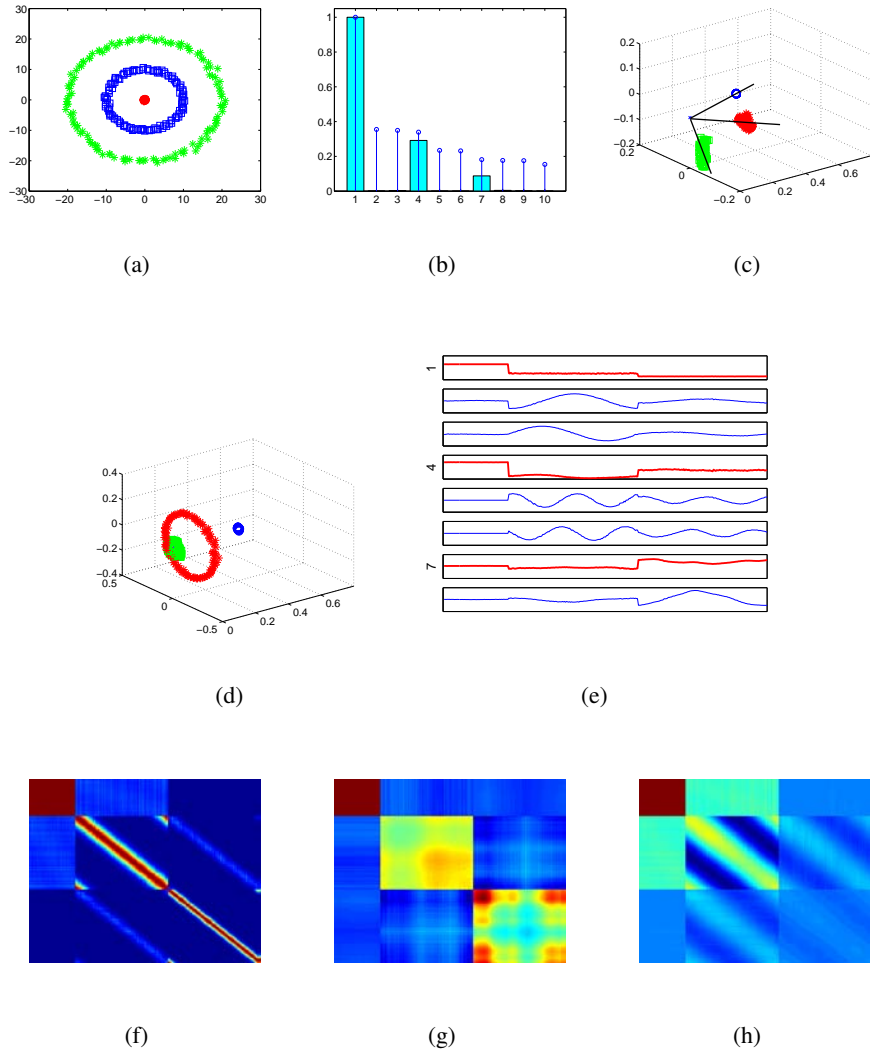

(a)

(b)

(c)

(d)

(e)

(f)

(g)

(h)

Figure 1: Examples of data transformations using kernel MaxEnt and kernel PCA.

with the state-of-the-art Ng et al. method (NG) [11] using the Laplacian matrix. The clustering is performed over a range of kernel sizes. The vertical line indicates the "optimal" kernel size using Silverman's rule. Over the whole range, kernel MaxEnt performs equally good as NG, and better than kernel PCA for small kernel sizes. Fig. 3 (b) shows a similar result for the data set shown in Fig. 1 (a). Kernel MaxEnt has the best performance over the most part of the kernel range. Fig. 3 (c) shows the mean result for the benchmark `thyroid` data set [13]. On this data set, kernel MaxEnt performs considerably better than the two other methods, over a wide range of kernel sizes.

These preliminary experiments show the potential benefits of kernel MaxEnt in data analysis, especially when the kernel space cost function is based on an angular measure. Using kernel MaxEnt makes the algorithm competitive to spectral clustering using the Laplacian matrix. We note that kernel MaxEnt in theory requires the full eigendecomposition, thus making it more computationally complex than clustering based on only the $C$ largest eigenvectors.

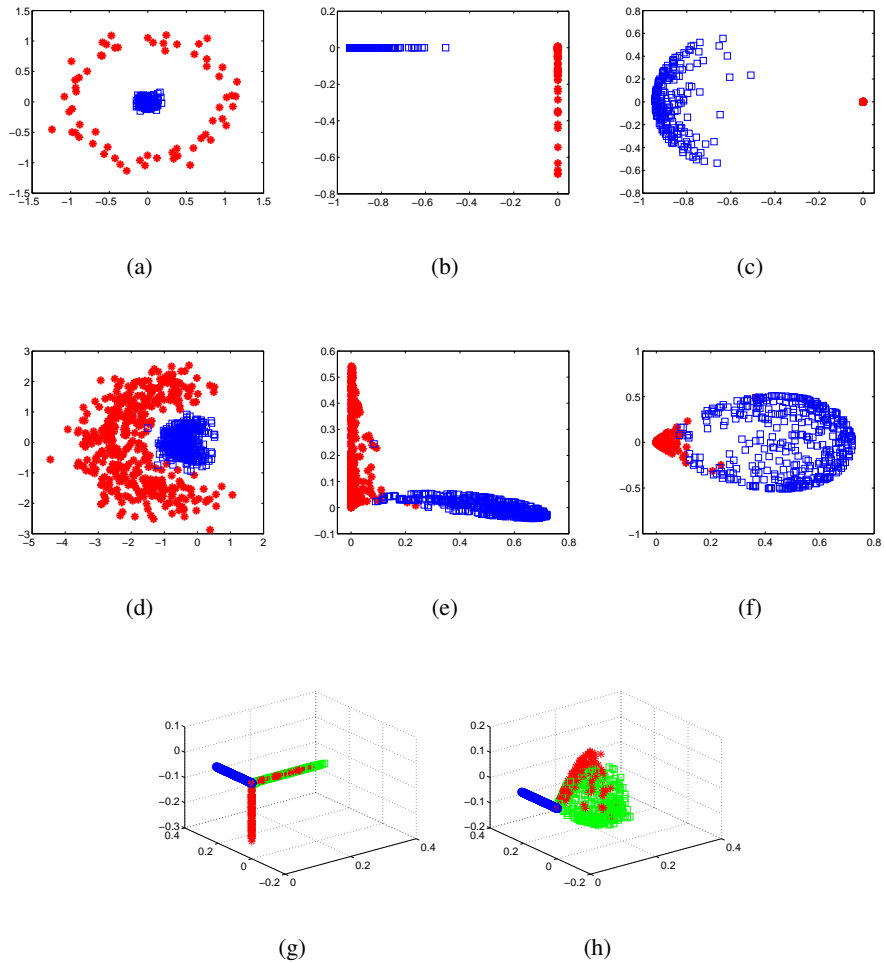

Figure 2: Examples of data transformations using kernel MaxEnt and kernel PCA.

## 6  Conclusions

In this paper, we have introduced a new data transformation technique, named kernel MaxEnt, which has a clear theoretical foundation based on the concept of maximum entropy preservation. The new method is similar in structure to kernel PCA, but may produce totally different transformed data sets. We have shown that kernel MaxEnt significantly enhances a recent spectral clustering algorithm. Kernel MaxEnt also produces a new kernel matrix, which may be useful for further data analysis.

Kernel MaxEnt requires the kernel to be a valid Parzen window (i.e. a density). Kernel PCA requires the kernel to be a Mercer kernel (positive semidefinite), hence not necessarily a density. In that sense, kernel PCA may use a broader class of kernels. On the other hand, kernel MaxEnt may use Parzen windows which are not Mercer kernels (indefinite), such as the Epanechnikov kernel. Kernel MaxEnt based on indefinite kernels will be studied in future work.

**Acknowledgements**
RJ is supported by NFR grant 171125/V30 and MG is supported by EPSRC grant EP/C010620/1.

## Footnotes

*Corresponding author. Phone: (+47) 776 46493. Email: robertj@phys.uit.no.

[1]In [3] the kernel feature space data was assumed centered, obtained by a centering operation of the kernel matrix. We do not assume centered data here.

## References

[1]  S. Roweis and L. Saul, "Nonlinear Dimensionality Reduction by Locally Linear Embedding," *Science*, vol. 290, pp. 2323–2326, 2000.

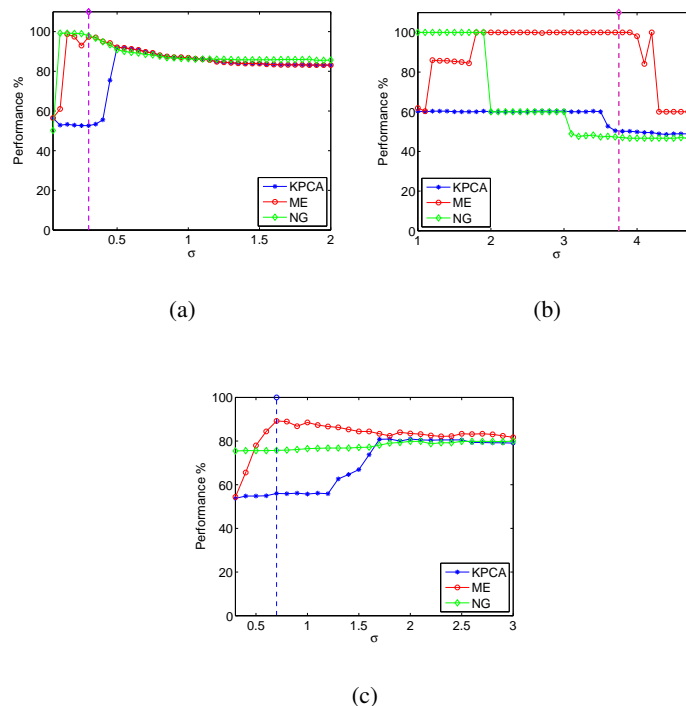

(a)                                    (b)

(c)

Figure 3: Clustering results.

[2]  J. Tenenbaum, V. de Silva, and J. C. Langford, "A Global Geometric Framework for Nonlinear Dimensionality Reduction," *Science*, vol. 290, pp. 2319–2323, 2000.

[3]  B. Schölkopf, A. J. Smola, and K. R. Müller, "Nonlinear Component Analysis as a Kernel Eigenvalue Problem," *Neural Computation*, vol. 10, pp. 1299–1319, 1998.

[4]  J. Shawe-Taylor and N. Cristianini, *Kernel Methods for Pattern Analysis*, Cambridge University Press, 2004.

[5]  R. Jenssen, D. Erdogmus, J. C. Principe, and T. Eltoft, "The Laplacian PDF Distance: A Cost Function for Clustering in a Kernel Feature Space," in *Advances in Neural Information Processing Systems 17*, MIT Press, Cambridge, 2005, pp. 625–632.

[6]  R. Jenssen, D. Erdogmus, J. C. Principe, and T. Eltoft, "Some Equivalences between Kernel Methods and Information Theoretic Methods," *Journal of VLSI Signal Processing, to appear*, 2006.

[7]  R. Jenssen, D. Erdogmus, J. C. Principe, and T. Eltoft, "Information Theoretic Angle-Based Spectral Clustering: A Theoretical Analysis and an Algorithm," in *Proceedings of International Joint Conference on Neural Networks*, Vancouver, Canada, July 16-21, 2006, pp. 4904–4911.

[8]  M. Girolami, "Orthogonal Series Density Estimation and the Kernel Eigenvalue Problem," *Neural Computation*, vol. 14, no. 3, pp. 669–688, 2002.

[9]  A. Renyi, "On Measures of Entropy and Information," *Selected Papers of Alfred Renyi, Akademiai Kiado, Budapest*, vol. 2, pp. 565–580, 1976.

[10]  E. Parzen, "On the Estimation of a Probability Density Function and the Mode," *The Annals of Mathematical Statistics*, vol. 32, pp. 1065–1076, 1962.

[11]  A. Y. Ng, M. Jordan, and Y. Weiss, "On Spectral Clustering: Analysis and an Algorithm," in *Advances in Neural Information Processing Systems, 14*, MIT Press, Cambridge, 2002, pp. 849–856.

[12]  B. W. Silverman, *Density Estimation for Statistics and Data Analysis*, Chapman and Hall, London, 1986.

[13]  G. Räetsch, T. Onoda, and K. R. Müller, "Soft Margins for Adaboost," *Machine Learning*, vol. 42, pp. 287–320, 2001.
